# Beyond Novelty Detection: Incongruent Events, when General and Specific Classifiers Disagree

### Abstract

Unexpected stimuli are a challenge to any machine learning algorithm. Here we identify distinct types of unexpected events, focusing on 'incongruent events' - when 'general level' and 'specific level' classifiers give conflicting predictions. We define a formal framework for the representation and processing of incongruent events: starting from the notion of label hierarchy, we show how partial order on labels can be deduced from such hierarchies. For each event, we compute its probability in different ways, based on adjacent levels (according to the partial order) in the label hierarchy. An incongruent event is an event where the probability computed based on some more specific level (in accordance with the partial order) is much smaller than the probability computed based on some more general level, leading to conflicting predictions. We derive algorithms to detect incongruent events from different types of hierarchies, corresponding to class membership or part membership. Respectively, we show promising results with real data on two specific problems: Out Of Vocabulary words in speech recognition, and the identification of a new sub-class (e.g., the face of a new individual) in audio-visual facial object recognition.

## 1 Introduction

Machine learning builds models of the world using training data from the application domain and prior knowledge about the problem. The models are later applied to future data in order to estimate the current state of the world. An implied assumption is that the future is stochastically similar to the past. The approach fails when the system encounters situations that are not anticipated from the past experience. In contrast, successful natural organisms identify new unanticipated stimuli and situations and frequently generate appropriate responses.

By definition, an unexpected event is one whose probability to confront the system is low, based on the data that has been observed previously. In line with this observation, much of the computational work on novelty detection focused on the probabilistic modeling of known classes, identifying outliers of these distributions as novel events (see e.g. [1, 2] for recent reviews). More recently, one-class classifiers have been proposed and used for novelty detection without the direct modeling of data distribution [3, 4]. There are many studies on novelty detection in biological systems [5], often focusing on regions of the hippocampus [6].

To advance beyond the detection of outliers, we observe that there are many different reasons why some stimuli could appear novel. Our work, presented in Section 2, focuses on unexpected events which are indicated by the incongruence between prediction induced by prior experience (training) and the evidence provided by the sensory data. To identify an item as incongruent, we use two parallel classifiers. One of them is strongly constrained by specific knowledge (both prior and data-derived), the other classifier is more general and less constrained. Both classifiers are assumed to yield class-posterior probability in response to a particular input signal. A sufficiently large discrepancy between posterior probabilities induced by input data in the two classifiers is taken as indication that an item is incongruent.

Thus, in comparison with most existing work on novelty detection, one new and important characteristic of our approach is that we look for a level of description where the novel event is highly probable. Rather than simply respond to an event which is rejected by all classifiers, which more often than not requires no special attention (as in pure noise), we construct and exploit a hierarchy of

representations. We attend to those events which are recognized (or accepted) at some more abstract levels of description in the hierarchy, while being rejected by the more concrete classifiers.

There are various ways to incorporate prior hierarchical knowledge and constraints within different classifier levels, as discussed in Section 3. One approach, used to detect images of unexpected incongruous objects, is to train the more general, less constrained classifier using a larger more diverse set of stimuli, e.g., the facial images of many individuals. The second classifier is trained using a more specific (i.e. smaller) set of specific objects (e.g., the set of Einstein's facial images). An incongruous item (e.g., a new individual) could then be identified by a smaller posterior probability estimated by the more specific classifiers relative to the probability from the more general classifier.

A different approach is used to identify unexpected (out-of-vocabulary) lexical items. The more general classifier is trained to classify sequentially speech sounds (phonemes) from a relatively short segments of the input speech signal (thus yielding an unconstrained sequence of phoneme labels); the more constrained classifier is trained to classify a particular set of words (highly constrained sequences of phoneme labels) from the information available in the whole speech sentence. A word that did not belong to the expected vocabulary of the more constrained recognizer could then be identified by discrepancy in posterior probabilities of phonemes derived from both classifiers.

Our second contribution in Section 2 is the presentation of a unifying theoretical framework for these two approaches. Specifically, we consider two kinds of hierarchies: *Part membership* as in biological taxonomy or speech, and *Class membership*, as in human categorization (or levels of categorization). We define a notion of partial order on such hierarchies, and identify those events whose probability as computed using different levels of the hierarchy does not agree. In particular, we are interested in those events that receive high probability at more general levels (for example, the system is certain that the new example is a dog), but low probability at more specific levels (in the same example, the system is certain that the new example is not any known dog breed). Such events correspond to many interesting situations that are worthy of special attention, including incongruous scenes and new sub-classes, as shown in Section 3.

## 2 Incongruent Events - unified approach

### 2.1 Introducing label hierarchy

The set of labels represents the knowledge base about stimuli, which is either given (by a teacher in supervised learning settings) or learned (in unsupervised or semi-supervised settings). In cognitive systems such knowledge is hardly ever a set; often, in fact, labels are given (or can be thought of) as a hierarchy. In general, hierarchies can be represented as directed graphs. The nodes of the graphs may be divided into distinct subsets that correspond to different entities (e.g., all objects that are animals); we call these subsets "levels". We identify two types of hierarchies:

**Part membership**, as in biological taxonomy or speech. For example, eyes, ears, and nose combine to form a head; head, legs and tail combine to form a dog.

**Class membership**, as in human categorization – where objects can be classified at different levels of generality, from sub-ordinate categories (most specific level), to basic level (intermediate level), to super-ordinate categories (most general level). For example, a Beagle (sub-ordinate category) is also a dog (basic level category), and it is also an animal (super-ordinate category).

The two hierarchies defined above induce constraints on the observed features in different ways. In the *class-membership* hierarchy, a parent class admits higher number of combinations of features than any of its children, i.e., the parent category is less constrained than its children classes. In contrast, a parent node in the *part-membership* hierarchy imposes stricter constraints on the observed features than a child node. This distinction is illustrated by the simple "toy" example shown in Fig. 1. Roughly speaking, in the class-membership hierarchy (right panel), the parent node is the *disjunction* of the child categories. In the part-membership hierarchy (left panel), the parent category represents a *conjunction* of the children categories. This difference in the effect of constraints between the two representations is, of course, reflected in the dependency of the posterior probability on the class, conditioned on the observations.

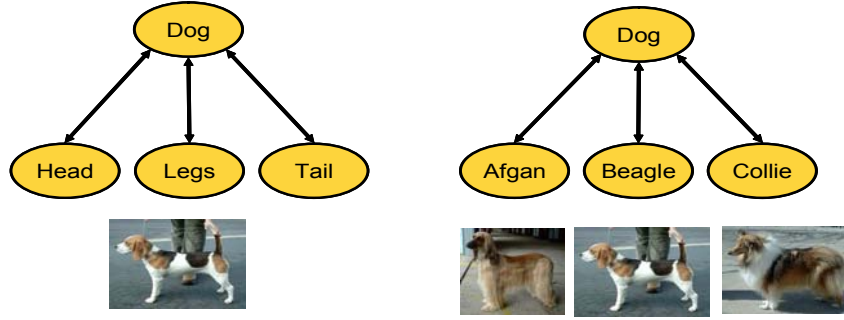

Figure 1: Examples. Left: *part-membership hierarchy,* the concept of a dog requires a conjunction of parts - a head, legs and tail. Right: *class-membership hierarchy,* the concept of a dog is defined as the disjunction of more specific concepts - Afghan, Beagle and Collie.

In order to treat different hierarchical representations uniformly we invoke the notion of partial order. Intuitively speaking, different levels in each hierarchy are related by a partial order: the more specific concept, which corresponds to a smaller set of events or objects in the world, is always smaller than the more general concept, which corresponds to a larger set of events or objects.

To illustrate this point, consider Fig. 1 again. For the part-membership hierarchy example (left panel), the concept of 'dog' requires a conjunction of parts as in $DOG = LEGS \cap HEAD \cap TAIL$, and therefore, for example, $DOG \subset LEGS \Rightarrow DOG \preceq LEGS$. Thus

$$DOG \preceq LEGS, \quad DOG \preceq HEAD, \quad DOG \preceq TAIL$$

In contrast, for the class-membership hierarchy (right panel), the class of dogs requires the conjunction of the individual members as in $DOG = AFGHAN \cup BEAGEL \cup COLLIE$, and therefore, for example, $DOG \supset AFGHAN \Rightarrow DOG \succeq AFGHAN$. Thus

$$DOG \succeq AFGHAN, \quad DOG \succeq BEAGEL, \quad DOG \succeq COLLIE$$

## 2.2 Definition of Incongruent Events

**Notations**

We assume that the data is represented as a Graph $\{G, E\}$ of Partial Orders ($GPO$). Each node in $G$ is a random variable which corresponds to a class or concept (or event). Each directed link in $E$ corresponds to partial order relationship as defined above, where there is a link from node $a$ to node $b$ iff $a \preceq b$.

For each node (concept) $a$, define $A^s = \{b \in G, \ b \preceq a\}$ - the set of all nodes (concepts) $b$ more specific (smaller) than $a$ in accordance with the given partial order; similarly, define $A^g = \{b \in G, \ a \preceq b\}$ - the set of all nodes (concepts) $b$ more general (larger) than $a$ in accordance with the given partial order.

For each concept $a$ and training data $\mathcal{T}$, we train up to 3 probabilistic models which are derived from $\mathcal{T}$ in different ways, in order to determine whether the concept $a$ is present in a new data point $X$:

- $Q_a(X)$: a probabilistic model of class $a$, derived from training data $\mathcal{T}$ without using the partial order relations in the $GPO$.

- If $|A^s| > 1$
  $Q_a^s(X)$: a probabilistic model of class $a$ which is based on the probability of concepts in $A^s$, assuming their independence of each other. Typically, the model incorporates some relatively simple conjunctive and/or disjunctive relations among concepts in $A^s$.

- If $|A^g| > 1$
  $Q_a^g(X)$: a probabilistic model of class $a$ which is based on the probability of concepts in $A^g$, assuming their independence of each other. Here too, the model typically incorporates some relatively simple conjunctive and/or disjunctive relations among concepts in $A^g$.

**Examples**

To illustrate, we use the simple examples shown in Fig. 1, where our concept of interest $a$ is the concept 'dog':

In the part-membership hierarchy (left panel), $|A^g| = 3$ (head, legs, tail). We can therefore learn 2 models for the class 'dog' ($Q^s_{\text{dog}}$ is not defined):

1. $Q_{\text{dog}}$ - obtained using training pictures of 'dogs' and 'not dogs' without body part labels.
2. $Q^g_{\text{dog}}$ - obtained using the outcome of models for head, legs and tail, which were trained on the same training set $\mathcal{T}$ with body part labels. For example, if we assume that concept $a$ is the conjunction of its part member concepts as defined above, and assuming that these part concepts are independent of each other, we get

$$Q^g_{\text{dog}} = \prod_{b \in A^g} Q_b = Q_{\text{Head}} \cdot Q_{\text{Legs}} \cdot Q_{\text{Tail}} \qquad (1)$$

In the class-membership hierarchy (right panel), $|A^s| = 3$ (Afghan, Beagle, Collie). If we further assume that a class-membership hierarchy is always a tree, then $|A^g| = 1$. We can therefore learn 2 models for the class 'dog' ($Q^g_{\text{dog}}$ is not defined):

1. $Q_{\text{dog}}$ - obtained using training pictures of 'dogs' and 'not dogs' without breed labels.
2. $Q^s_{\text{dog}}$ - obtained using the outcome of models for Afghan, Beagle and Collie, which were trained on the same training set $\mathcal{T}$ with only specific dog type labels. For example, if we assume that concept $a$ is the disjunction of its sub-class concepts as defined above, and assuming that these sub-class concepts are independent of each other, we get

$$Q^s_{\text{dog}} = \sum_{b \in A^s} Q_b = Q_{Afghan} + Q_{Beagle} + Q_{Collie}$$

**Incongruent events**

In general, we expect the different models to provide roughly the same probability for the presence of concept $a$ in data $X$. A mismatch between the predictions of the different models should raise the red flag, possibly indicating that something new and interesting had been observed. In particular, we are interested in the following discrepancy:

***Definition***: *Observation $X$ is* incongruent *if there exists a concept $'a'$ such that*

$$Q^g_a(X) \gg Q_a(X) \text{ or } Q_a(X) \gg Q^s_a(X). \qquad (2)$$

Alternatively, observation $X$ is *incongruent* if a discrepancy exists between the inference of the two classifiers: either the classifier based on the more general descriptions from level $g$ accepts the $X$ while the direct classier rejects it, or the direct classifier accepts $X$ while the classifier based on the more specific descriptions from level $s$ rejects it. In either case, the concept receives high probability at the more general level (according to the $GPO$), but much lower probability when relying only on the more specific level.

Let us discuss again the examples we have seen before, to illustrate why this definition indeed captures interesting "surprises":

- In the part-membership hierarchy (left panel of Fig. 1), we have

$$Q^g_{\text{dog}} = Q_{\text{Head}} \cdot Q_{\text{Legs}} \cdot Q_{\text{Tail}} \gg Q_{\text{dog}}$$

  In other words, while the probability of each part is high (since the multiplication of those probabilities is high), the 'dog' classifier is rather uncertain about the existence of a dog in this data.

  How can this happen? Maybe the parts are configured in an unusual arrangement for a dog (as in a 3-legged cat), or maybe we encounter a donkey with a cat's tail (as in Shrek 3). Those are two examples of the kind of unexpected events we are interested in.

- In the class-membership hierarchy (right panel of Fig. 1), we have

$$Q_{\text{dog}}^s = Q_{\text{A}fghan} + Q_{\text{Beagle}} + Q_{\text{Collie}} \ll Q_{\text{dog}}$$

In other words, while the probability of each sub-class is low (since the sum of these probabilities is low), the 'dog' classifier is certain about the existence of a dog in this data.

How may such a discrepancy arise? Maybe we are seeing a new type of dog that we haven't seen before - a Pointer. The dog model, if correctly capturing the notion of 'dogness', should be able to identify this new object, while models of previously seen dog breeds (Afghan, Beagle and Collie) correctly fail to recognize the new object.

## 3   Incongruent events: algorithms

Our definition for incongruent events in the previous section is indeed unified, but as a result quite abstract. In this section we discuss two different algorithmic implementations, one generative and one discriminative, which were developed for the *part membership* and *class membership* hierarchies respectively (see definition in Section 1). In both cases, we use the notation $Q(x)$ for the class probability as defined above, and $p(x)$ for the estimated probability.

### 3.1   *Part membership* - a generative algorithm

Consider the left panel of Fig. 1. The event in the top node is incongruent if its probability is low, while the probability of all its descendants is high.

In many applications, such as speech recognition, one computes the probability of events (sentences) based on a generative model (corresponding to a specific language) which includes a dictionary of parts (words). At the top level the event probability is computed conditional on the model; in which case typically the parts are assumed to be independent, and the event probability is computed as the multiplication of the parts probabilities conditioned on the model. For example, in speech processing and assuming a specific language (e.g., English), the probability of the sentence is typically computed by multiplying the probability of each word using an HMM model trained on sentences from a specific language. At the bottom level, the probability of each part is computed independently of the generative model.

More formally, Consider an event $u$ composed of parts $w_k$. Using the generative model of events and assuming the conditional independence of the parts given this model, the prior probability of the event is given by the product of prior probabilities of the parts,

$$p(u|L) = \prod_k p(w_k|L) \tag{3}$$

where $L$ denotes the generative model (e.g., the language).

For measurement $X$, we compute $Q(X)$ as follows

$$Q(X) = p(X|L) = \sum_u p(X|u, L)p(u|L) \approx p(X|\bar{u}, L)p(\bar{u}|L) = p(X|\bar{u}) \prod_k p(w_k|L) \tag{4}$$

using $p(X|u, L) = p(X|u)$ and (3), and where $\bar{u} = \arg\max_u\ p(u|L)$ is the most likely interpretation. At the risk of notation abuse, $\{w_k\}$ now denote the parts which compose the most likely event $\bar{u}$. We assume that the first sum is dominated by the maximal term.

Given a part-membership hierarchy, we can use (1) to compute the probability $Q^g(X)$ directly, without using the generative model $L$.

$$Q^g(X) = p(X) = \sum_u p(X|u)p(u) \geq p(X|\bar{u})p(\bar{u}) = p(X|\bar{u}) \prod_k p(w_k) \tag{5}$$

It follows from (4) and (5) that

$$\frac{Q(X)}{Q^g(X)} \leq \prod_k \frac{p(w_k|L)}{p(w_k)} \tag{6}$$

We can now conclude that $X$ is an incongruent event according to our definition if there exists at least one part $k$ in the final event $\bar{u}$, such that $p(w_k) \gg p(w_k|L)$ (assuming all other parts have roughly the same conditional and unconditional probabilities). In speech processing, a sentence is incongruent if it includes an incongruent word - a word whose probability based on the generative language model is low, but whose direct probability (not constrained by the language model) is high.

**Example: Out Of Vocabulary (OOV) words**

For the detection of OOV words, we performed experiments using a Large Vocabulary Continuous Speech Recognition (LVCSR) system on the Wall Street Journal Corpus (WSJ). The evaluation set consists of 2.5 hours. To introduce OOV words, the vocabulary was restricted to the 4968 most frequent words from the language training texts, leaving the remaining words unknown to the model. A more detailed description is given in [7].

In this task, we have shown that the comparison between two parallel classifiers, based on strong and weak posterior streams, is effective for the detection of OOV words, and also for the detection of recognition errors. Specifically, we use the derivation above to detect out of vocabulary words, by comparing their probability when computed based on the language model, and when computed based on mere acoustic modeling. The best performance was obtained by the system when a Neural Network (NN) classifier was used for the direct estimation of frame-based OOV scores. The network was directly fed by posteriors from the strong and the weak systems. For the WSJ task, we achieved performance of around 11% Equal-Error-Rate (EER) (Miss/False Alarm probability), see Fig. 2.

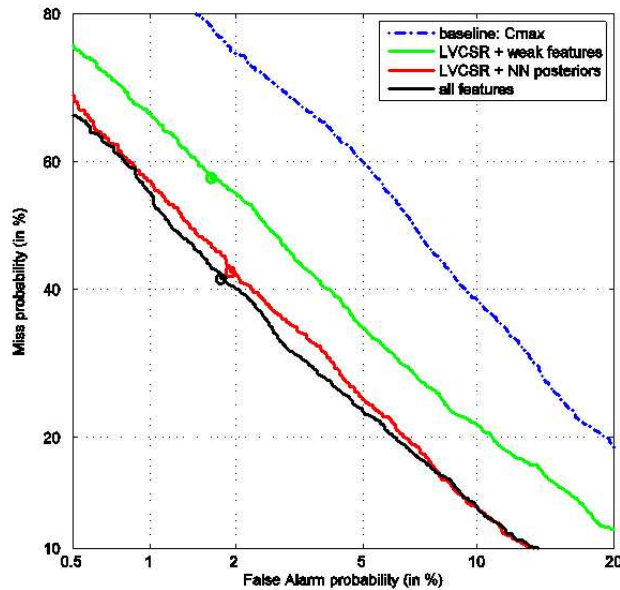

Figure 2: Several techniques used to detect OOV: (i) Cmax: Confidence measure computed ONLY from strongly constrained Large Vocabulary Continuous Speech Recognizer (LVCSR), with frame-based posteriors. (ii) LVCSR+weak features: Strongly and weakly constrained recognizers, compared via the KL-divergence metric. (iii) LVCSR+NN posteriors: Combination of strong and weak phoneme posteriors using NN classifier. (iv) all features: fusion of (ii) and (iii) together.

## 3.2  *Class membership* - a discriminative algorithm

Consider the right panel of Fig. 1. The general class in the top node is incongruent if its probability is high, while the probability of all its sub-classes is low. In other words, the classifier of the parent object accepts the new observation, but all the children object classifiers reject it. Brute force computation of this definition may follow the path taken by traditional approaches to novelty detection, e.g., looking for rejection by all one class classifiers corresponding to sub-class objects.

The result we have obtained by this method were mediocre, probably because generative models are not well suited for the task. Instead, it seems like discriminative classifiers, trained to discriminate

between objects at the sub-class level, could be more successful. We note that unlike traditional approaches to novelty detection, which must use generative models or one-class classifiers in the absence of appropriate discriminative data, our dependence on object hierarchy provides discriminative data as a by-product. In other words, after the recognition by a parent-node classifier, we may use classifiers trained to discriminate between its children to implement a discriminative novelty detection algorithm.

Specifically, we used the approach described in [8] to build a unified representation for all objects in the sub-class level, which is the representation computed for the parent object whose classifier had accepted (positively recognized) the object. In this feature space, we build a classifier for each sub-class based on the majority vote between pairwise discriminative classifiers. Based on these classifiers, each example (accepted by the parent classifier) is assigned to one of the sub-classes, and the average margin over classifiers which agree with the final assignment is calculated. The final classifier then uses a threshold on this average margin to identify each object as known sub-class or new sub-class. Previous research in the area of face identification can be viewed as an implicit use of this propsed framework, see e.g. [9].

**Example: new face recognition from audio-visual data**

We tested our algorithm on audio-visual speaker verification. In this setup, the general parent category level is the 'speech' (audio) and 'face' (visual), and the different individuals are the offspring (sub-class) levels. The task is to identify an individual as belonging to the trusted group of individuals vs. being unknown, i.e. known sub-class vs. new sub-class in a class membership hierarchy.

The unified representation of the visual cues was built using the approach described in [8]. All objects in the sub-class level (different individuals) were represented using the representation learnt for the parent level ('face'). For the audio cues we used the Perceptual linear predictive (PLP) Cepstral features [10] as the unified representation. We used SVM classifiers with RBF kernel as the pairwise discriminative classifiers for each of the different audio/visual representations separately.

Data was collected for our experiments using a wearable device, which included stereo panoramic vision sensors and microphone arrays. In the recorded scenario, individuals walked towards the device and then read aloud an identical text; we acquired 30 sequences with 17 speakers (see Fig. 3 for an example). We tested our method by choosing six speakers as members of the trusted group, while the rest were assumed unknown.

The method was applied separately using each one of the different modalities, and also in an integrated manner using both modalities. For this fusion the audio signal and visual signal were synchronized, and the winning classification margins of both signals were normalized to the same scale and averaged to obtain a single margin for the combined method.

Since the goal is to identify novel incongruent events, true positive and false positive rates were calculated by considering all frames from the unknown test sequences as positive events and the known individual test sequences as negative events. We compared our method to novelty detection based on one-class SVM [3] extended to our multi-class case. Decision was obtained by comparing the maximal margin over all one-class classifiers to a varying threshold. As can be seen in Fig. 3, our method performs substantially better in both modalities as compared to the "standard" one class approach for novelty detection. Performance is further improved by fusing both modalities.

## 4   Summary

Unexpected events are typically identified by their low posterior probability. In this paper we employed label hierarchy to obtain a few probability values for each event, which allowed us to tease apart different types of unexpected events. In general there are 4 possibilities, based on the classifiers' response at two adjacent levels:

|   | Specific level | General level | possible reason |
|---|---|---|---|
| 1 | reject | reject | noisy measurements, or a totally new concept |
| 2 | reject | accept | incongruent concept |
| 3 | accept | reject | inconsistent with partial order, models are wrong |
| 4 | accept | accept | known concept |

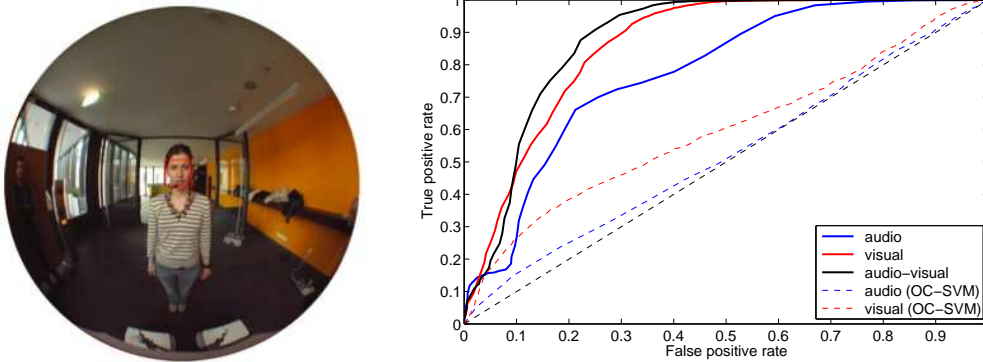

Figure 3: Left: Example: one frame used for the visual verification task. Right: True Positive vs. False Positive rates when detecting unknown vs. trusted individuals. The unknown are regarded as positive events. Results are shown for the proposed method using both modalities separately and the combined method (solid lines). For comparison, we show results with a more traditional novelty detection method using One Class SVM (dashed lines).

We focused above on the second type of events - incongruent concepts, which have not been studied previously in isolation. Such events are characterized by some discrepancy between the response of two classifiers, which can occur for a number different reasons: *Context:* in a given context such as the English language, a sentence containing a Czech word is assigned low probability. In the visual domain, in a given context such as a street scene, otherwise high probability events such as "car" and "elephant" are not likely to appear together. *New sub-class:* a new object has been encountered, of some known generic type but unknown specifics.

We described how our approach can be used to design new algorithms to address these problems, showing promising results on real speech and audio-visual facial datasets.

## References

[1] Markou, M., Singh, S.: Novelty detection: a review-part 1: statistical approaches. Signal Processing **83** (2003) 2499 – 2521

[2] Markou, M., Singh, S.: Novelty detection: a review-part 2: neural network based approaches. Signal Processing **83** (2003) 2481–2497

[3] Scholkopf, B., Williamson, R.C., Smola, A.J., Shawe-Taylor, J., Platt, J.: Support vector method for novelty detection. In: Proc. NIPS. Volume 12. (2000) 582–588

[4] Lanckrietand, G.R.G., Ghaoui, L.E., Jordan, M.I.: Robust novelty detection with single-class mpm. In: Proc. NIPS. Volume 15. (2003) 929–936

[5] Berns, G.S., Cohen, J.D., Mintun, M.A.: Brain regions responsive to novelty in the absence of awareness. Science **276** (1997) 1272 – 1275

[6] Rokers, B., Mercado, E., Allen, M.T., Myers, C.E., Gluck, M.A.: A connectionist model of septohip-pocampal dynamics during conditioning: Closing the loop. Behavioral Neuroscience **116** (2002) 48–62

[7] Burget, L., Schwarz, P., Matejka, P., Hannemann, M., Rastrow, A., White, C., Khudanpur, S., Hermansky, H., Cernocky, J.: Combination of strongly and weakly constrained recognizers for reliable detection of oovs. In: Proceedings of IEEE Int. Conf. on Acoustics, Speech, and Signal Processing (ICASSP). (2008)

[8] Bar-Hillel, A., Weinshall, D.: Subordinate class recognition using relational object models. Proc. NIPS **19** (2006)

[9] Lanitis, A., Taylor, C.J., Cootes, T.F.: A unified approach to coding and interpreting face images. In: Proc. ICCV. (1995) 368–373

[10] Hermansky, H.: Perceptual linear predictive (PLP) analysis of speech. The Journal of the Acoustical Society of America **87** (1990) 1738

